# Kernels for Structured Natural Language Data

**Jun Suzuki, Yutaka Sasaki, and Eisaku Maeda**
NTT Communication Science Laboratories, NTT Corp.
2-4 Hikaridai, Seika-cho, Soraku-gun, Kyoto, 619-0237 Japan
{jun, sasaki, maeda}@cslab.kecl.ntt.co.jp

## Abstract

This paper devises a novel kernel function for structured natural language data. In the field of Natural Language Processing, feature extraction consists of the following two steps: (1) syntactically and semantically analyzing raw data, i.e., character strings, then representing the results as discrete structures, such as parse trees and dependency graphs with part-of-speech tags; (2) creating (possibly high-dimensional) numerical feature vectors from the discrete structures. The new kernels, called Hierarchical Directed Acyclic Graph (HDAG) kernels, directly accept DAGs whose nodes can contain DAGs. HDAG data structures are needed to fully reflect the syntactic and semantic structures that natural language data inherently have. In this paper, we define the kernel function and show how it permits efficient calculation. Experiments demonstrate that the proposed kernels are superior to existing kernel functions, e.g., sequence kernels, tree kernels, and bag-of-words kernels.

## 1 Introduction

Recent developments in kernel technology enable us to handle discrete structures, such as sequences, trees, and graphs. Kernel functions suitable for *Natural Language Processing (NLP)* have recently been proposed. *Convolution Kernels* [4, 12] demonstrate how to build kernels over discrete structures. Since texts can be analyzed as discrete structures, these discrete kernels have been applied to NLP tasks, such as sequence kernels [8, 9] for text categorization and tree kernels [1, 2] for (shallow) parsing.

In this paper, we focus on tasks in the application areas of NLP, such as Machine Translation, Text Summarization, Text Categorization and Question Answering. In these tasks, richer types of information within texts, such as syntactic and semantic information, are required for higher performance. However, syntactic information and semantic information are formed by very complex structures that cannot be written in simple structures, such as sequences and trees. The motivation of this paper is to propose kernels specifically suited to structured natural language data. The proposed kernels can handle several of the structures found within texts and calculate kernels with regard to these structures at a practical cost and time. Accordingly, these kernels can be efficiently applied to learning and clustering problems in NLP applications.

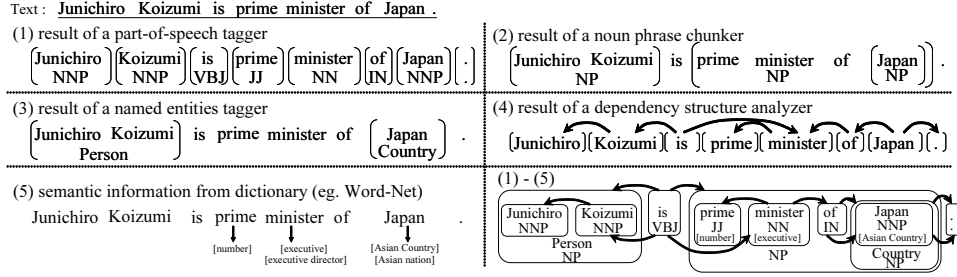

Figure 1: Examples of structures within texts as determined by basic NLP tools

## 2 Structured Natural Language Data for Application Tasks in NLP

In general, natural language data contain many kinds of syntactic and semantic structures. For example, texts have several levels of syntactic and semantic chunks, such as part-of-speech (POS) chunks, named entities (NEs), noun phrase (NP) chunks, sentences, and discourse segments, and these are bound by relation structures, such as dependency structures, anaphora, discourse relations and coreference. These syntactic and semantic structures can provide important information for understanding natural language and, moreover, tackling real tasks in application areas of NLP. The accuracies of basic NLP tools such as POS taggers, NP chunkers, NE taggers, and dependency structure analyzers have improved to the point that they can help to develop real applications.

This paper proposes a method to handle these syntactic and semantic structures in a single framework: We combine the results of basic NLP tools to make one hierarchically structured data set. Figure 1 shows an example of structures within texts analyzed by basic NLP tools that are currently available and that offer easy use and high performance. As shown in Figure 1, structures in texts can be hierarchical or recursive "graphs in graph". A certain node can be constructed or characterized by other graphs. Nodes usually have several kinds of attributes, such as words, POS tags, semantic information such as WordNet [3], and classes of the named entities. Moreover, the relations between nodes are usually directed. Therefore, we should employ a (1) directed, (2) multi-labeled, and (3) hierarchically structured graph to model structured natural language data.

Let $V$ be a set of vertices (or nodes) and $E$ be a set of edges (or links). Then, a graph $G = (V, E)$ is called a *directed graph* if $E$ is a set of directed links $E \subset V \times V$.

**Definition 1** *(Multi-Labeled Graph) Let $\Gamma$ be a set of labels (or attributes) and $M \subset V \times \Gamma$ be label allocations. Then, $G = (V, E, M)$ is called a multi-labeled graph.*

**Definition 2** *(Hierarchically Structured Graph) Let $\mathcal{G}_i = (V_i, E_i)$ be a subgraph in $\mathcal{G} = (V, E)$ where $V_i \subseteq V$ and $E_i \subseteq E$, and $\mathbb{G} = \{\mathcal{G}_1, \ldots, \mathcal{G}_n\}$ be a set of subgraphs in $\mathcal{G}$. $F \subset V \times \mathbb{G}$ represents a set of vertical links from a node $v \in V$ to a subgraph $\mathcal{G}_i \in \mathbb{G}$. Then, $\mathcal{G} = (V, E, \mathbb{G}, F)$ is called a hierarchically structured graph if each node has at most one vertical edge. Intuitively, vertical link $f_{i,\mathcal{G}_j} \in F$ from node $v_i$ to graph $\mathcal{G}_j$ indicates that node $v_i$ contains graph $\mathcal{G}_j$.*

Finally, in this paper, we successfully represent structured natural language data by using a *multi-labeled hierarchical directed graph*.

**Definition 3** *(Multi-Labeled Hierarchical Directed Graph) $\mathcal{G} = (V, E, M, \mathbb{G}, F)$ is a multi-labeled hierarchical directed graph.*

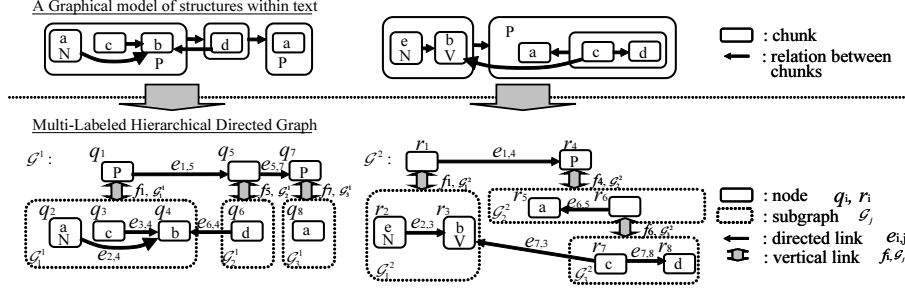

Figure 2: Examples of Hierarchical Directed Graph structures (these are also HDAG): each letter represents an attribute

Figure 2 shows examples of multi-labeled hierarchical directed graphs. In this paper, we call a multi-labeled hierarchical directed graph a hierarchical directed graph.

## 3 Kernels on Hierarchical Directed Acyclic Graph

At first, in order to calculate kernels efficiently, we add one constraint: that the hierarchical directed graph has no cyclic paths. First, we define a path on a Hierarchical Directed Graph. If a node has no vertical link, then the node is called a *terminal node*, which is denoted as $T \subset V$; otherwise it is a *non-terminal node*, which is denoted as $\bar{T} \subset V$.

**Definition 4** *(Hierarchical Path (HiP)) Let $p = \langle v_i, e_{i,j}, v_j, \ldots, v_k, e_{k,l}, v_l \rangle$ be a path. Let $\Upsilon(v)$ be a function that returns a subgraph $\mathcal{G}_i$ that is linked with $v$ by a vertical link if $v \in \bar{T}$. Let $\mathcal{P}(\mathcal{G})$ be a function that returns the set of all HiPs in $\mathcal{G}$, where links between $v \in \mathcal{G}$ and $v \notin \mathcal{G}$ are ignored. Then, $p^h = \langle h(v_i), e_{i,j}, h(v_j), \ldots, h(v_k), e_{k,l}, h(v_l) \rangle$ is defined as a HiP, where $h(v)$ returns $vp_x^h$, $p_x^h \in \mathcal{P}(\mathcal{G}_x)$ s.t. $\mathcal{G}_x = \Upsilon(v)$ if $v \in \bar{T}$ otherwise returns $v$. Intuitively, a HiP is constructed by a path in the path structure, e.g., $p^h = \langle v_i, e_{i,j}, v_j \langle v_m, e_{m,n}, v_n \rangle, \ldots, v_k, e_{k,l}, v_l \rangle$.*

**Definition 5** *(Hierarchical Directed Acyclic Graph (HDAG)) hierarchical directed graph $\mathcal{G} = (V, E, M, \mathbb{G}, F)$ is an HDAG if there is no HiP from any node $v$ to the same node $v$.*

A primitive feature for defining kernels on HDAGs is a *hierarchical attribute subsequence*.

**Definition 6** *(Hierarchical Attribute Subsequence (HiAS)) A HiAS is defined as a list of attributes with hierarchical information extracted from nodes on HiPs.*

For example, let $p^h = \langle v_i, e_{i,j}, v_j \langle v_m, e_{m,n}, v_n \rangle, \ldots, v_k, e_{k,l}, v_l \rangle$ be a HiP, then, HiASs in $p^h$ are written as $\tau(p^h) = \langle a_i, a_j \langle a_m, a_n \rangle, \ldots, a_k, a_l \rangle$, which is all combinations for all $a_i \in \tau(v_i)$, where $\tau(v)$ of node $v$ is a function that returns the set of attributes allocated to node $v$, and $\tau(p^h)$ of HiP $p^h$ is a function that returns all possible HiASs extracted from HiP $p^h$.

$\Gamma^*$ denotes all possible HiASs constructed by the attribute in $\Gamma$ and $\gamma_i \in \Gamma^*$ denotes the $i$'th HiAS. An explicit representation of a feature vector of an HDAG kernel is defined as $\phi(\mathcal{G}) = (\phi_1(\mathcal{G}), \ldots, \phi_{|\Gamma^*|}(\mathcal{G}))$, where $\phi$ represents the explicit feature mapping from HDAG to the numerical feature space. The value of $\phi_i(\mathcal{G})$ becomes the weighted number of occurrences of $\gamma_i$ in $\mathcal{G}$. According to this approach, the HDAG kernel, $K(\mathcal{G}^1, \mathcal{G}^2) = \sum_{i=1}^{|\Gamma^*|} \langle \phi_i(\mathcal{G}^1) \cdot \phi_i(\mathcal{G}^2) \rangle$, calculates the inner product of the weighted common HiASs in

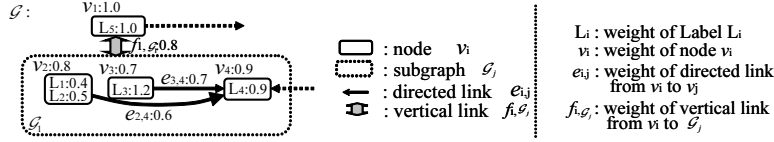

Figure 3: An Example of Hierarchical Directed Graph "$\mathcal{G}$" with weight factors

two HDAGs, $\mathcal{G}^1$ and $\mathcal{G}^2$. In this paper, we use | stand for the meaning of "such that," since it is simple.

$$K_{\text{HDAG}}(\mathcal{G}^1, \mathcal{G}^2) = \sum_{\gamma_i \in \Gamma^*} \sum_{\gamma_i \in \tau(p_1^h) | p_1^h \in \mathcal{P}(\mathcal{G}^1)} \sum_{\gamma_i \in \tau(p_2^h) | p_2^h \in \mathcal{P}(\mathcal{G}^2)} \mathcal{W}_{\gamma_i}(p_1^h) \mathcal{W}_{\gamma_i}(p_2^h), \quad (1)$$

where $\mathcal{W}_{\gamma_i}(p^h)$ represents the weight value of HiAS $\gamma_i$ in HiP $p^h$. The weight of HiAS $\gamma_i$ in HiP $p^h$ is determined by

$$\mathcal{W}_{\gamma_i}(p^h) = \prod_{v \in V(p^h)} W_V(v) \prod_{e_{i,j} \in E(p^h)} W_E(v_i, v_j) \prod_{f_{i,\mathcal{G}_j} \in F(p^h)} W_F(v_i, \mathcal{G}_j) \prod_{a \in \tau(\gamma_i)} W_\Gamma(a), \quad (2)$$

where $W_V(v)$, $W_E(v_i, v_j)$, $W_F(v_i, \mathcal{G}_j)$, and $W_\Gamma(a)$ represent the weight of node $v$, link from $v_i$ to $v_j$, vertical link from $v_i$ to subgraph $\mathcal{G}_j$, and attribute $a$, respectively. An example of how each weight factor is given is shown in Figure 3. In the case of NL data, for example, $W_\Gamma(a)$ might be given by the score of $tf * idf$ from large scale documents, $W_V(v)$ by the type of chunk such as word, phrase or named entity, $W_E(v_i, v_j)$ by the type of relation between $v_i$ and $v_j$, and $W_F(v_i, \mathcal{G}_j)$ by the number of nodes in $\mathcal{G}_j$.

**Soft Structural Matching Frameworks**

Since HDAG kernels permit not only the exact matching of substructures but also approximate matching, we add the framework of *node skip* and *relaxation of hierarchical information*.

First, we discuss the framework of the *node skip*. We introduce decay function $\Lambda_V(v)(0 < \Lambda_V(v) \leq 1)$, which represents the cost of skipping node $v$ when extracting HiASs from the HiPs, which is almost the same architecture as [8]. For example, a HiAS under the node skips is written as $\langle *\langle a_2, a_3 \rangle, *, \langle a_5 \rangle \rangle$ from HiP $\langle v_1 \langle v_2, v_3 \rangle, v_4, \langle v_5 \rangle \rangle$, where $*$ is the explicit representation of a node that is skipped.

Next, in the case of the relaxation of hierarchical information, we perform two processes: (1) we form one hierarchy if there is multiple hierarchy information in the same point, for example, $\langle \langle \langle a_i, a_j \rangle \rangle, a_k \rangle$ becomes $\langle \langle a_i, a_j \rangle, a_k \rangle$; and (2) we delete hierarchical information if there exists only one node, for example, $\langle \langle a_i \rangle, a_j, a_k \rangle$ becomes $\langle a_i, a_j, a_k \rangle$.

These two frameworks achieve approximate substructure matching automatically. Table 1 shows an explicit representation of the common HiASs (features) of $\mathcal{G}^1$ and $\mathcal{G}^2$ in Figure 2. For the sake of simplicity, for all the weights $W_V(v)$, $W_E(v_i, v_j)$, $W_F(v_i, \mathcal{G}_j)$, and $W_\Gamma(a)$, are taken as 1 and for all $v$, $\Lambda_V(v) = \lambda$ if $v$ has at least one attribute, otherwise $\Lambda_V(v) = 1$.

**Efficient Recursive Computation**

In general, when the dimension of the feature space $|\Gamma^*|$ becomes very high, it is computationally infeasible to generate feature vector $\phi(\mathcal{G})$ explicitly. We define an efficient calculation formula between HDAGs $\mathcal{G}^1$ and $\mathcal{G}^2$, which is written as:

$$K_{\text{HDAG}}(\mathcal{G}^1, \mathcal{G}^2) = \sum_{q \in Q} \sum_{r \in R} K(q, r), \quad (3)$$

Table 1: Common HiASs of $\mathcal{G}^1$ and $\mathcal{G}^2$ in Figure 2: (N.S. represents the node skip, H.R. represents the relaxation of hierarchical information)

| $\mathcal{G}^1$ | | | $\mathcal{G}^2$ | | | N.S. | | N.S.+ H.R. | |
|---|---|---|---|---|---|---|---|---|---|
| HiAS with $*$ | HiAS | value | HiAS with $*$ | HiAS | value | common HiAS | value | common HiAS | value |
| $\langle P \rangle$ | $\langle P \rangle$ | 2 | $\langle P \rangle$ | $\langle P \rangle$ | 1 | $\langle P \rangle$ | 2 | $\langle P \rangle$ | 2 |
| $\langle N \rangle$ | $\langle N \rangle$ | 1 | $\langle N \rangle$ | $\langle N \rangle$ | 1 | $\langle N \rangle$ | 1 | $\langle N \rangle$ | 1 |
| $\langle a \rangle$ | $\langle a \rangle$ | 2 | $\langle a \rangle$ | $\langle a \rangle$ | 1 | $\langle a \rangle$ | 2 | $\langle a \rangle$ | 2 |
| $\langle b \rangle$ | $\langle b \rangle$ | 1 | $\langle b \rangle$ | $\langle b \rangle$ | 1 | $\langle b \rangle$ | 1 | $\langle b \rangle$ | 1 |
| $\langle c \rangle$ | $\langle c \rangle$ | 1 | $\langle c \rangle$ | $\langle c \rangle$ | 1 | $\langle c \rangle$ | 1 | $\langle c \rangle$ | 1 |
| $\langle d \rangle$ | $\langle d \rangle$ | 1 | $\langle d \rangle$ | $\langle d \rangle$ | 1 | $\langle d \rangle$ | 1 | $\langle d \rangle$ | 1 |
| $\langle c,b \rangle$ | $\langle c,b \rangle$ | 1 | $\langle c,b \rangle$ | $\langle c,b \rangle$ | 1 | $\langle c,b \rangle$ | 1 | $\langle c,b \rangle$ | 1 |
| $\langle d,b \rangle$ | $\langle d,b \rangle$ | 1 | $\langle\langle d\rangle, \langle\langle b\rangle\rangle\rangle$ | $\langle\langle d\rangle, \langle\langle b\rangle\rangle\rangle$ | 1 | - | 0 | $\langle b,d \rangle$ | 1 |
| $P\langle a \rangle$ | $P\langle a \rangle$ | 2 | $P\langle a \rangle$ | $P\langle a \rangle$ | 1 | $P\langle a \rangle$ | 2 | $P\langle a \rangle$ | 2 |
| $P\langle c \rangle$ | $P\langle c \rangle$ | 1 | $P\langle\langle c\rangle\rangle$ | $P\langle\langle c\rangle\rangle$ | 1 | - | 0 | $P\langle c \rangle$ | 1 |
| $\langle *\langle N\rangle, \langle *\rangle, *\langle a\rangle\rangle$ | $\langle\langle N\rangle, \langle a\rangle\rangle$ | $\lambda^3$ | $\langle\langle N\rangle, *\langle a\rangle\rangle$ | $\langle\langle N\rangle, \langle a\rangle\rangle$ | $\lambda$ | $\langle\langle N\rangle, \langle a\rangle\rangle$ | $\lambda^4$ | $\langle N,a \rangle$ | $\lambda^4$ |
| $\langle *\langle N\rangle, \langle *\rangle, P\rangle$ | $\langle\langle N\rangle, P\rangle$ | $\lambda^2$ | $\langle\langle N\rangle, P\rangle$ | $\langle\langle N\rangle, P\rangle$ | 1 | $\langle\langle N\rangle, P\rangle$ | $\lambda^2$ | $\langle N,P \rangle$ | $\lambda^2$ |
| $\langle N,b \rangle$ | $\langle N,b \rangle$ | 1 | $\langle N,b \rangle$ | $\langle N,b \rangle$ | 1 | $\langle N,b \rangle$ | 1 | $\langle N,b \rangle$ | 1 |
| $\langle *\langle N\rangle, \langle d\rangle\rangle$ | $\langle\langle N\rangle, \langle d\rangle\rangle$ | $\lambda$ | $\langle\langle N\rangle, *\langle\langle d\rangle\rangle\rangle$ | $\langle\langle N\rangle, \langle\langle d\rangle\rangle\rangle$ | $\lambda$ | - | 0 | $\langle N,d \rangle$ | $\lambda^2$ |
| $\langle *\langle b\rangle, \langle *\rangle, *\langle a\rangle\rangle$ | $\langle\langle b\rangle, \langle a\rangle\rangle$ | $\lambda^3$ | $\langle\langle b\rangle, *\langle a\rangle\rangle$ | $\langle\langle b\rangle, \langle a\rangle\rangle$ | $\lambda$ | $\langle\langle b\rangle, \langle a\rangle\rangle$ | $\lambda^4$ | $\langle b,a \rangle$ | $\lambda^4$ |
| $\langle *\langle b\rangle, \langle *\rangle, P\rangle$ | $\langle\langle b\rangle, P\rangle$ | $\lambda^2$ | $\langle\langle b\rangle, P\rangle$ | $\langle\langle b\rangle, P\rangle$ | 1 | $\langle\langle b\rangle, P\rangle$ | $\lambda^2$ | $\langle b,P \rangle$ | $\lambda^2$ |
| $\langle *\langle b\rangle, \langle d\rangle\rangle$ | $\langle\langle b\rangle, \langle d\rangle\rangle$ | $\lambda$ | $\langle\langle b\rangle, *\langle\langle d\rangle\rangle\rangle$ | $\langle\langle b\rangle, \langle\langle d\rangle\rangle\rangle$ | $\lambda$ | - | 0 | $\langle b,d \rangle$ | $\lambda^2$ |
| $\langle *\langle c\rangle, \langle *\rangle, *\langle a\rangle\rangle$ | $\langle\langle c\rangle, \langle a\rangle\rangle$ | $\lambda^3$ | $\langle\langle c\rangle, a\rangle$ | $\langle\langle c\rangle, a\rangle$ | 1 | - | 0 | $\langle c,a \rangle$ | $\lambda^3$ |
| $\langle *\langle c\rangle, \langle d\rangle\rangle$ | $\langle\langle c\rangle, \langle d\rangle\rangle$ | $\lambda$ | $\langle c,d \rangle$ | $\langle c,d \rangle$ | 1 | - | 0 | $\langle c,d \rangle$ | $\lambda$ |
| $\langle\langle d\rangle, *\langle a\rangle\rangle$ | $\langle\langle d\rangle, \langle a\rangle\rangle$ | $\lambda$ | $\langle\langle d\rangle, a\rangle$ | $\langle\langle d\rangle, a\rangle$ | 1 | - | 0 | $\langle d,a \rangle$ | $\lambda$ |
| $\langle *\langle N\rangle, \langle *\rangle, P\langle a\rangle\rangle$ | $\langle\langle N\rangle, P\langle a\rangle\rangle$ | $\lambda^2$ | $\langle\langle N\rangle, P\langle a\rangle\rangle$ | $\langle\langle N\rangle, P\langle a\rangle\rangle$ | 1 | $\langle\langle N\rangle, P\langle a\rangle\rangle$ | $\lambda^2$ | $\langle N, P\langle a\rangle\rangle$ | $\lambda^2$ |
| $\langle *\langle b\rangle, \langle *\rangle, P\langle a\rangle\rangle$ | $\langle\langle b\rangle, P\langle a\rangle\rangle$ | $\lambda^2$ | $\langle\langle b\rangle, P\langle a\rangle\rangle$ | $\langle\langle b\rangle, P\langle a\rangle\rangle$ | 1 | $\langle\langle b\rangle, P\langle a\rangle\rangle$ | $\lambda^2$ | $\langle b, P\langle a\rangle\rangle$ | $\lambda^2$ |
| $\langle *\langle N,b\rangle, \langle *\rangle, *\langle a\rangle\rangle$ | $\langle\langle N,b\rangle, \langle a\rangle\rangle$ | $\lambda^3$ | $\langle\langle N,b\rangle, *\langle a\rangle\rangle$ | $\langle\langle N,b\rangle, \langle a\rangle\rangle$ | $\lambda$ | $\langle\langle N,b\rangle, \langle a\rangle\rangle$ | $\lambda^4$ | $\langle\langle N,b\rangle, a\rangle$ | $\lambda^4$ |
| $\langle *\langle N,b\rangle, \langle *\rangle, P\rangle$ | $\langle\langle N,b\rangle, P\rangle$ | $\lambda^2$ | $\langle\langle N,b\rangle, P\rangle$ | $\langle\langle N,b\rangle, P\rangle$ | 1 | $\langle\langle N,b\rangle, P\rangle$ | $\lambda^2$ | $\langle\langle N,b\rangle, P\rangle$ | $\lambda^2$ |
| $\langle *\langle N,b\rangle, \langle d\rangle\rangle$ | $\langle\langle N,b\rangle, \langle d\rangle\rangle$ | $\lambda$ | $\langle\langle N,b\rangle, *\langle\langle d\rangle\rangle\rangle$ | $\langle\langle N,b\rangle, \langle\langle d\rangle\rangle\rangle$ | $\lambda$ | - | 0 | $\langle\langle N,b\rangle, d\rangle$ | $\lambda^2$ |
| $\langle *\langle N,b\rangle, \langle *\rangle, P\langle a\rangle\rangle$ | $\langle\langle N,b\rangle, P\langle a\rangle\rangle$ | $\lambda^2$ | $\langle\langle N,b\rangle, P\langle a\rangle\rangle$ | $\langle\langle N,b\rangle, P\langle a\rangle\rangle$ | 1 | $\langle\langle N,b\rangle, P\langle a\rangle\rangle$ | $\lambda^2$ | $\langle\langle N,b\rangle, P\langle a\rangle\rangle$ | $\lambda^2$ |

where $Q = \{q_1, \ldots, q_{|Q|}\}$ and $R = \{r_1, \ldots, r_{|R|}\}$ represent nodes in $\mathcal{G}^1$ and $\mathcal{G}^2$, respectively. $K(q,r)$ represents the sum of the weighted common HiASs that are extracted from the HiPs whose sink nodes are $q$ and $r$.

$$K(q,r) = J''_{\mathcal{G}^1,\mathcal{G}^2}(q,r)H(q,r) + \hat{H}(q,r)I(q,r) + I(q,r) \qquad (4)$$

Function $I(q,r)$ returns the weighted number of common attributes of nodes $q$ and $r$,

$$I(q,r) = W_V(q)W_V(r) \sum_{a_1 \in \tau(q)} \sum_{a_2 \in \tau(r)} W_\Gamma(a_1)W_\Gamma(a_2)\delta(a_1, a_2), \qquad (5)$$

where $\delta(a_1, a_2) = 1$ if $a_1 = a_2$, and 0 otherwise. Let $H(q,r)$ be a function that returns the sum of the weighted common HiASs between $q$ and $r$ including $\Upsilon(q)$ and $\Upsilon(r)$.

$$H(q,r) = \begin{cases} I(q,r) + \left(I(q,r) + \Lambda_V(q)\Lambda_V(r)\right)\hat{H}(q,r), & \text{if } q,r \in \bar{T} \\ I(q,r), & \text{otherwise} \end{cases} \qquad (6)$$

$$\hat{H}(q,r) = \sum_{s \in \mathcal{G}_i^1 | \mathcal{G}_i^1 = \Upsilon(q)} \sum_{t \in \mathcal{G}_j^2 | \mathcal{G}_j^2 = \Upsilon(r)} W_F(q, \mathcal{G}_i^1)W_F(r, \mathcal{G}_j^2)J_{\mathcal{G}_i^1, \mathcal{G}_j^2}(s,t) \qquad (7)$$

Let $J_{x,y}(q,r)$, $J'_{x,y}(q,r)$, and $J''_{x,y}(q,r)$, where $x,y$ are (sub)graphs, be recursive functions to calculate $H(q,r)$ and $K(q,r)$.

$$J_{x,y}(q,r) = J''_{x,y}(q,r)H(q,r) + H(q,r) \qquad (8)$$

$$J'_{x,y}(q,r) = \begin{cases} \sum_{t \in \{\psi(r) \cap V(y)\}} W_E(q,t)\left(\Lambda'_V(t)J'_{x,y}(q,t) + J_{x,y}(q,t)\right), & \text{if } \psi(r) \neq \emptyset \\ 0, & \text{otherwise} \end{cases} \qquad (9)$$

$$J''_{x,y}(q,r) = \begin{cases} \sum_{s \in \{\psi(q) \cap V(x)\}} W_E(s,r)\left(\Lambda'_V(s)J''_{x,y}(s,r) + J'_{x,y}(s,r)\right), & \text{if } \psi(q) \neq \emptyset \\ 0, & \text{otherwise} \end{cases} \qquad (10)$$

where $\Lambda'_V(v) = \Lambda_V(v) \prod_{t \in \mathcal{G}_i | \mathcal{G}_i = \Upsilon(v)} \Lambda_V(t)$ if $v \in \bar{T}$, $\Lambda'_V(v) = \Lambda_V(v)$ otherwise. Function $\psi(q)$ returns a set of nodes that have direct links to node $q$. $\psi(q) = \emptyset$ means that no node has a direct link to $s$.

Next, we show the formula when using the framework of relaxation of hierarchical information. The functions have the same meanings as in the previous formula. We denote $\tilde{H}(q,r) = H(q,r) + H'(q,r)$.

$$K(q,r) = J''_{\mathcal{G}^1, \mathcal{G}^2}(q,r)\tilde{H}(q,r) + \big(H'(q,r) + H''(q,r)\big)I(q,r) + I(q,r) \tag{11}$$

$$H(q,r) = \big(H'(q,r) + H''(q,r)\big)I(q,r) + H''(q,r) + I(q,r) \tag{12}$$

$$H'(q,r) = \begin{cases} \displaystyle\sum_{t \in \mathcal{G}_j^2 | \mathcal{G}_j^2 = \Upsilon(r)} W_F(r, \mathcal{G}_j^2)\tilde{H}(q,t), & \text{if } r \in \bar{T} \\ 0, & \text{otherwise} \end{cases} \tag{13}$$

$$H''(q,r) = \begin{cases} \displaystyle\sum_{s \in \mathcal{G}_i^1 | \mathcal{G}_i^1 = \Upsilon(q)} W_F(q, \mathcal{G}_i^1)H(s,r) + \hat{H}(q,r), & \text{if } q,r \in \bar{T} \\ \displaystyle\sum_{s \in \mathcal{G}_i^1 | \mathcal{G}_i^1 = \Upsilon(q)} W_F(q, \mathcal{G}_i^1)H(s,r), & \text{if } q \in \bar{T} \\ 0, & \text{otherwise} \end{cases} \tag{14}$$

$$J_{x,y}(q,r) = J''_{x,y}(q,r)\tilde{H}(q,r) \tag{15}$$

$$J'_{x,y}(q,r) = \begin{cases} \displaystyle\sum_{t \in \{\psi(r) \cap V(y)\}} W_E(q,t)\big(\Lambda'_V(t)J'_{x,y}(q,t) + J_{x,y}(q,t) + \tilde{H}(q,t)\big), & \text{if } \psi(r) \neq \emptyset \\ 0, & \text{otherwise} \end{cases} \tag{16}$$

Functions $I(q,r)$, $J''_{x,y}(q,r)$, and $\hat{H}(q,r)$ are the same as those shown above.

According to equation (3), given the recursive definition of $K_{\text{HDAG}}(q,r)$, the value between two HDAGs can be calculated in time $O(|Q||R|)$. In actual use, we may want to evaluate only the subset of all HiASs whose sizes are under $n$ when determining the kernel value because of the problem discussed in [1]. This can simply realized by not calculating those HiASs whose size exceeds $n$ when calculating $K(q,r)$; the calculation cost becomes $O(n|Q||R|)$.

Finally, we normalize the values of the HDAG kernels to remove any bias introduced by the number of nodes in the graphs. This normalization corresponds to the standard unit norm normalization of examples in the feature space corresponding to the kernel space $\hat{K}(x,y) = K(x,y) \cdot (K(x,x)K(y,y))^{-1/2}$ [4].

We will now elucidate an efficient processing algorithm. First, as a pre-process, the nodes are sorted under two conditions, $V(\Upsilon(v)) \prec v$ and $\Psi(v) \prec v$, where $\bar{\Psi}(v)$ represents all nodes that have a path to $v$. The dynamic programming technique can be used to compute HDAG kernels very efficiently: By following the sorted order, the values that are needed to calculate $K(q,r)$ have already been calculated in the previous calculation.

## 4 Experiments

Our aim was to test the efficiency of using the richer syntactic and semantic structures available within texts, which can be treated now for the first time by our proposed method. We evaluated the performance of the proposed method in the actual NLP task of Question Classification, which is similar to the Text Classification task except that it requires many

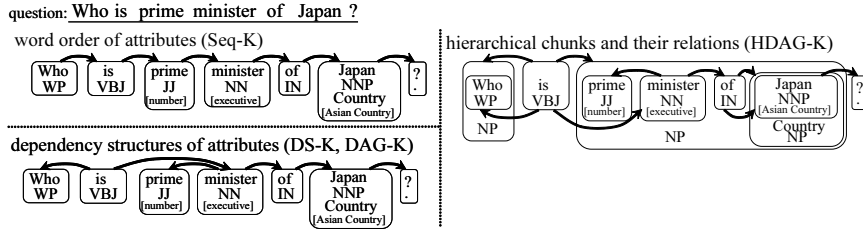

Figure 4: Examples of input data of comparison methods

Table 2: Results of question classification by SVM with comparison kernel functions evaluated by F-measure

| | TIME_TOP | | | | LOCATION | | | | ORGANIZATION | | | | NUMEX | | | |
|---|---|---|---|---|---|---|---|---|---|---|---|---|---|---|---|---|
| $n$ | 1 | 2 | 3 | 4 | 1 | 2 | 3 | 4 | 1 | 2 | 3 | 4 | 1 | 2 | 3 | 4 |
| **HDAG-K** | - | **.951** | .942 | .926 | - | .802 | **.813** | .784 | - | **.716** | .712 | .697 | - | .916 | **.922** | .874 |
| DAG-K | - | .946 | .913 | .869 | - | .803 | .774 | .729 | - | .704 | .671 | .610 | - | .912 | .880 | .813 |
| DS-K | - | .615 | .564 | .403 | - | .544 | .507 | .466 | - | .535 | .509 | .419 | - | .602 | .504 | .424 |
| Seq-K | - | .946 | .910 | .866 | - | .792 | .774 | .733 | - | .706 | .668 | .595 | - | .913 | .885 | .815 |
| BOW-K | 899 | .906 | .885 | .853 | .748 | .772 | .757 | .745 | .638 | .690 | .633 | .571 | .841 | .846 | .804 | .719 |

more semantic features within texts [7, 10]. We used three different QA data sets written in Japanese [10].

We compared the performance of the proposed kernel, the HDAG Kernel (HDAG-K), with DAG kernels (DAG-K), Dependency Structure kernels (DS-K) [2], and sequence kernels (Seq-K) [9]. Moreover, we evaluated the *bag-of-words* kernel (BOW-K) [6], that is, the bag-of-words with polynomial kernels, as the baseline method. The main difference between each method is the ability to treat syntactic and semantic information within texts. Figure 4 shows the differences of input objects between each method. For better understanding, these examples are shown in English. We used words, named entity tags, and semantic information [5] for attributes. Seq-K only treats word order, DS-K and DAG-K treat dependency structures, and HDAG-K treats the NP and NE chunks with their dependency structures. We used the same formula with our proposed method for DAG-K. Comparing HDAG-K to DAG-K shows the difference in performance between handling the hierarchical structures and not handling them. We extended Seq-K and DS-K to improve the total performance and to establish a more equal evaluation, with the same conditions, against our proposed method. Note that though DAG-K and DS-K handle input objects of the same form, their kernel calculation methods differ as do their return values. We used node skip parameter $\Lambda_V(v) = 0.5$ for all nodes $v$ in each comparison.

We used SVM [11] as a kernel-based machine learning algorithm. We evaluated the performance of the comparison methods with question type *TIME_TOP*, *ORGANIZATION*, *LOCATION*, and *NUMEX*, which are defined in the CRL QA-data[1].

Table 2 shows the average F-measure as evaluated by 5-fold cross validation. $n$ in Table 2 indicates the threshold of an attribute's number, that is, we evaluated only those HiASs that contain less than $n$-attributes for each kernel calculation. As shown in this table, HDAG-K showed the best performance in the experiments. The experiments in this paper were designed to investigate how to improve the performance by using the richer syntactic and semantic structures within texts. In the task of Question Classification, a given question is classified into Question Type, which reflects the intention of the question. These results

indicate that our approach, incorporating richer structure features within texts, is well suited to the tasks in the NLP applications.

The original DS-K requires exact matching of the tree structure, even when it is extended for more flexible matching. This is why DS-K showed the worst performance in our experiments. The sequence, DAG, and HDAG kernels offer approximate matching by the framework of node skip, which produces better performance in the tasks that evaluate the intention of the texts.

The structure of HDAG approaches that of DAG if we do not consider the hierarchical structure. In addition, the structures of sequences and trees are entirely included in that of DAG. Thus, the HDAG kernel subsumes some of the discrete kernels, such as sequence, tree, and graph kernels.

## 5  Conclusions

This paper proposed HDAG kernels, which can handle more of the rich syntactic and semantic information present within texts. Our proposed method is a very generalized framework for handling structured natural language data. We evaluated the performance of HDAG kernels with the real NLP task of question classification. Our experiments showed that HDAG kernels offer better performance than sequence kernels, tree kernels, and the baseline method bag-of-words kernels if the target task requires the use of the richer information within texts.

## Footnotes

[1]http://www.cs.nyu.edu/~sekine/PROJECT/CRLQA/

## References

[1] M. Collins and N. Duffy. Convolution Kernels for Natural Language. In *Proc. of Neural Information Processing Systems (NIPS'2001)*, 2001.

[2] M. Collins and N. Duffy. Parsing with a Single Neuron: Convolution Kernels for Natural Language Problems. In *Technical Report UCS-CRL-01-10*. UC Santa Cruz, 2001.

[3] C. Fellbaum. *WordNet: An Electronic Lexical Database*. MIT Press, 1998.

[4] D. Haussler. Convolution Kernels on Discrete Structures. In *Technical Report UCS-CRL-99-10*. UC Santa Cruz, 1999.

[5] S. Ikehara, M. Miyazaki, S. Shirai, A. Yokoo, H. Nakaiwa, K. Ogura, Y. Oyama, and Y. Hayashi, editors. *The Semantic Attribute System,* Goi-Taikei — A Japanese Lexicon, volume 1. Iwanami Publishing, 1997. (in Japanese).

[6] T. Joachims. Text Categorization with Support Vector Machines: Learning with Many Relevant Features. In *Proc. of European Conference on Machine Learning(ECML '98)*, pages 137–142, 1998.

[7] X. Li and D. Roth. Learning Question Classifiers. In *Proc. of the 19th International Conference on Computational Linguistics (COLING 2002)*, pages 556–562, 2002.

[8] H. Lodhi, C. Saunders, J. Shawe-Taylor, N. Cristianini, and C. Watkins. Text Classification Using String Kernel. *Journal of Machine Learning Research*, 2:419–444, 2002.

[9] N. Cancedda and E. Gaussier and C. Goutte and J.-M. Renders. Word-Sequence Kernels. *Journal of Machine Learning Research*, 3:1059–1082, 2003.

[10] J. Suzuki, H. Taira, Y. Sasaki, and E. Maeda. Question Classification using HDAG Kernel. In *Workshop on Multilingual Summarization and Question Answering (2003)*, pages 61–68, 2003.

[11] V. N. Vapnik. *The Nature of Statistical Learning Theory*. Springer, 1995.

[12] C. Watkins. Dynamic Alignment Kernels. In *Technical Report CSD-TR-98-11*. Royal Holloway, University of London Computer Science Department, 1999.
